# Lower Bounds on the Complexity of Approximating Continuous Functions by Sigmoidal Neural Networks

**Michael Schmitt**
Lehrstuhl Mathematik und Informatik
Fakultät für Mathematik
Ruhr-Universität Bochum
D–44780 Bochum, Germany
*mschmitt@lmi.ruhr-uni-bochum.de*

## Abstract

We calculate lower bounds on the size of sigmoidal neural networks that approximate continuous functions. In particular, we show that for the approximation of polynomials the network size has to grow as $\Omega((\log k)^{1/4})$ where $k$ is the degree of the polynomials. This bound is valid for any input dimension, i.e. independently of the number of variables. The result is obtained by introducing a new method employing upper bounds on the Vapnik-Chervonenkis dimension for proving lower bounds on the size of networks that approximate continuous functions.

## 1 Introduction

Sigmoidal neural networks are known to be universal approximators. This is one of the theoretical results most frequently cited to justify the use of sigmoidal neural networks in applications. By this statement one refers to the fact that sigmoidal neural networks have been shown to be able to approximate any continuous function arbitrarily well. Numerous results in the literature have established variants of this universal approximation property by considering distinct function classes to be approximated by network architectures using different types of neural activation functions with respect to various approximation criteria, see for instance [1, 2, 3, 5, 6, 11, 12, 14, 15]. (See in particular Scarselli and Tsoi [15] for a recent survey and further references.)

All these results and many others not referenced here, some of them being constructive, some being merely existence proofs, provide upper bounds for the network size asserting that good approximation is possible if there are sufficiently many network nodes available. This, however, is only a partial answer to the question that mainly arises in practical applications: "Given some function, how many network nodes are needed to approximate it?" Not much attention has been focused on establishing lower bounds on the network size and, in particular, for the approximation of functions over the reals. As far as the computation of binary-valued

functions by sigmoidal networks is concerned (where the output value of a network is thresholded to yield 0 or 1) there are a few results in this direction. For a specific Boolean function Koiran [9] showed that networks using the standard sigmoid $\sigma(y) = 1/(1 + e^{-y})$ as activation function must have size $\Omega(n^{1/4})$ where $n$ is the number of inputs. (When measuring network size we do not count the input nodes here and in what follows.) Maass [13] established a larger lower bound by constructing a binary-valued function over $\mathbb{R}^n$ and showing that standard sigmoidal networks require $\Omega(n)$ many network nodes for computing this function. The first work on the complexity of sigmoidal networks for approximating continuous functions is due to DasGupta and Schnitger [4]. They showed that the standard sigmoid in network nodes can be replaced by other types of activation functions without increasing the size of the network by more than a polynomial. This yields indirect lower bounds for the size of sigmoidal networks in terms of other network types. DasGupta and Schnitger [4] also claimed the size bound $A^{\Omega(1/d)}$ for sigmoidal networks with $d$ layers approximating the function $\sin(Ax)$.

In this paper we consider the problem of using the standard sigmoid $\sigma(y) = 1/(1 + e^{-y})$ in neural networks for the approximation of polynomials. We show that at least $\Omega((\log k)^{1/4})$ network nodes are required to approximate polynomials of degree $k$ with small error in the $l_\infty$ norm. This bound is valid for arbitrary input dimension, i.e., it does not depend on the number of variables. (Lower bounds can also be obtained from the results on binary-valued functions mentioned above by interpolating the corresponding functions by polynomials. This, however, requires growing input dimension and does not yield a lower bound in terms of the degree.) Further, the bound established here holds for networks of any number of layers. As far as we know this is the first lower bound result for the approximation of polynomials. From the computational point of view this is a very simple class of functions; they can be computed using the basic operations addition and multiplication only. Polynomials also play an important role in approximation theory since they are dense in the class of continuous functions and some approximation results for neural networks rely on the approximability of polynomials by sigmoidal networks (see, e.g., [2, 15]).

We obtain the result by introducing a new method that employs upper bounds on the Vapnik-Chervonenkis dimension of neural networks to establish lower bounds on the network size. The first use of the Vapnik-Chervonenkis dimension to obtain a lower bound is due to Koiran [9] who calculated the above-mentioned bound on the size of sigmoidal networks for a Boolean function. Koiran's method was further developed and extended by Maass [13] using a similar argument but another combinatorial dimension. Both papers derived lower bounds for the computation of binary-valued functions (Koiran [9] for inputs from $\{0, 1\}^n$, Maass [13] for inputs from $\mathbb{R}^n$). Here, we present a new technique to show that and how lower bounds can be obtained for networks that approximate continuous functions. It rests on two fundamental results about the Vapnik-Chervonenkis dimension of neural networks. On the one hand, we use constructions provided by Koiran and Sontag [10] to build networks that have large Vapnik-Chervonenkis dimension and consist of gates that compute certain arithmetic functions. On the other hand, we follow the lines of reasoning of Karpinski and Macintyre [7] to derive an upper bound for the Vapnik-Chervonenkis dimension of these networks from the estimates of Khovanskiĭ [8] and a result due to Warren [16].

In the following section we give the definitions of sigmoidal networks and the Vapnik-Chervonenkis dimension. Then we present the lower bound result for function approximation. Finally, we conclude with some discussion and open questions.

## 2   Sigmoidal Neural Networks and VC Dimension

We briefly recall the definitions of a sigmoidal neural network and the Vapnik-Chervonenkis dimension (see, e.g., [7, 10]). We consider *feedforward neural networks* which have a certain number of input nodes and one output node. The nodes which are not input nodes are called *computation nodes* and associated with each of them is a real number $t$, the *threshold*. Further, each edge is labelled with a real number $w$ called *weight*. Computation in the network takes place as follows: The input values are assigned to the input nodes. Each computation node applies the standard sigmoid $\sigma(y) = 1/(1 + e^{-y})$ to the sum $w_1 x_1 + \cdots + w_r x_r - t$ where $x_1, \ldots, x_r$ are the values computed by the node's predecessors, $w_1, \ldots, w_r$ are the weights of the corresponding edges, and $t$ is the threshold. The output value of the network is defined to be the value computed by the output node. As it is common for approximation results by means of neural networks, we assume that the output node is a linear gate, i.e., it just outputs the sum $w_1 x_1 + \cdots + w_r x_r - t$. (Clearly, for computing functions on finite sets with output range $[0, 1]$ the output node may apply the standard sigmoid as well.) Since $\sigma$ is the only sigmoidal function that we consider here we will refer to such networks as *sigmoidal neural networks*. (Sigmoidal functions in general need to satisfy much weaker assumptions than $\sigma$ does.) The definition naturally generalizes to networks employing other types of gates that we will make use of (e.g. linear, multiplication, and division gates).

The Vapnik-Chervonenkis dimension is a combinatorial dimension of a function class and is defined as follows: A *dichotomy* of a set $S \subseteq \mathbb{R}^n$ is a partition of $S$ into two disjoint subsets $(S_0, S_1)$ such that $S_0 \cup S_1 = S$. Given a set $\mathcal{F}$ of functions mapping $\mathbb{R}^n$ to $\{0, 1\}$ and a dichotomy $(S_0, S_1)$ of $S$, we say that $\mathcal{F}$ *induces* the dichotomy $(S_0, S_1)$ on $S$ if there is some $f \in \mathcal{F}$ such that $f(S_0) \subseteq \{0\}$ and $f(S_1) \subseteq \{1\}$. We say further that $\mathcal{F}$ *shatters* $S$ if $\mathcal{F}$ induces all dichotomies on $S$. The *Vapnik-Chervonenkis (VC) dimension* of $\mathcal{F}$, denoted $\mathrm{VCdim}(\mathcal{F})$, is defined as the largest number $m$ such that there is a set of $m$ elements that is shattered by $\mathcal{F}$. We refer to the VC dimension of a neural network, which is given in terms of a "feedforward architecture", i.e. a directed acyclic graph, as the VC dimension of the class of functions obtained by assigning real numbers to all its programmable parameters, which are in general the weights and thresholds of the network or a subset thereof. Further, we assume that the output value of the network is thresholded at $1/2$ to obtain binary values.

## 3   Lower Bounds on Network Size

Before we present the lower bound on the size of sigmoidal networks required for the approximation of polynomials we first give a brief outline of the proof idea. We will define a sequence of univariate polynomials $(p_n)_{n \geq 1}$ by means of which we show how to construct neural architectures $\mathcal{N}_n$ consisting of various types of gates such as linear, multiplication, and division gates, and, in particular, gates that compute some of the polynomials. Further, this architecture has a single weight as programmable parameter (all other weights and thresholds are fixed). We then demonstrate that, assuming the gates computing the polynomials can be approximated by sigmoidal neural networks sufficiently well, the architecture $\mathcal{N}_n$ can shatter a certain set by assigning suitable values to its programmable weight. The final step is to reason along the lines of Karpinski and Macintyre [7] to obtain via Khovanskiĭ's estimates [8] and Warren's result [16] an upper bound on the VC dimension of $\mathcal{N}_n$ in terms of the number of its computation nodes. (Note that we cannot directly apply Theorem 7 of [7] since it does not deal with division gates.) Comparing this bound with the cardinality of the shattered set we will then be able

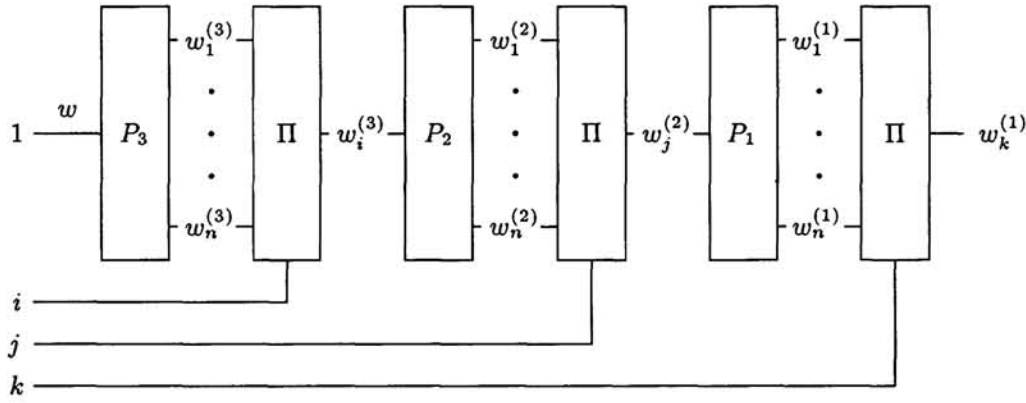

Figure 1: The network $\mathcal{N}_n$ with values $k, j, i, 1$ assigned to the input nodes $x_1, x_2, x_3, x_4$ respectively. The weight $w$ is the only programmable parameter of the network.

to conclude with a lower bound on the number of computation nodes in $\mathcal{N}_n$ and thus in the networks that approximate the polynomials.

Let the sequence $(p_n)_{n \geq 1}$ of polynomials over $\mathbb{R}$ be inductively defined by

$$p_n(x) = \begin{cases} 4x(1-x) & n = 1 \ , \\ p(p_{n-1}(x)) & n \geq 2 \ . \end{cases}$$

Clearly, this uniquely defines $p_n$ for every $n \geq 1$ and it can readily be seen that $p_n$ has degree $2^n$. The main lower bound result is made precise in the following statement.

**Theorem 1** *Sigmoidal neural networks that approximate the polynomials $(p_n)_{n \geq 1}$ on the interval $[0,1]$ with error at most $O(2^{-n})$ in the $l_\infty$ norm must have at least $\Omega(n^{1/4})$ computation nodes.*

**Proof.** For each $n$ a neural architecture $\mathcal{N}_n$ can be constructed as follows: The network has four input nodes $x_1, x_2, x_3, x_4$. Figure 1 shows the network with input values assigned to the input nodes in the order $x_4 = 1, x_3 = i, x_2 = j, x_1 = k$. There is one weight which we consider as the (only) programmable parameter of $\mathcal{N}_n$. It is associated with the edge outgoing from input node $x_4$ and is denoted by $w$. The computation nodes are partitioned into six levels as indicated by the boxes in Figure 1. Each level is itself a network. Let us first assume, for the sake of simplicity, that all computations over real numbers are exact. There are three levels labeled with $\Pi$, having $n + 1$ input nodes and one output node each, that compute so-called projections $\pi : \mathbb{R}^{n+1} \to \mathbb{R}$ where $\pi(y_1, \ldots, y_n, a) = y_a$ for $a \in \{1, \ldots, n\}$.

The levels labeled $P_3, P_2, P_1$ have one input node and $n$ output nodes each. Level $P_3$ receives the constant 1 as input and thus the value $w$ which is the parameter of the network. We define the output values of level $P_\lambda$ for $\lambda = 3, 2, 1$ by

$$w_b^{(\lambda)} = p_{b \cdot n^{\lambda-1}}(v) \ , \quad b = 1, \ldots, n$$

where $v$ denotes the input value to level $P_\lambda$. This value is equal to $w$ for $\lambda = 3$ and $\pi(w_1^{(\lambda+1)}, \ldots, w_n^{(\lambda+1)}, x_{\lambda+1})$ otherwise. We observe that $w_{b+1}^{(\lambda)}$ can be calculated from

$w_b^{(\lambda)}$ as $p_{n^{\lambda-1}}(w_b^{(\lambda)})$. Therefore, the computations of level $P_\lambda$ can be implemented using $n$ gates each of them computing the function $p_{n^{\lambda-1}}$.

We show now that $\mathcal{N}_n$ can shatter a set of cardinality $n^3$. Let $S = \{1, \ldots, n\}^3$. It has been shown in Lemma 2 of [10] that for each $(\beta_1, \ldots, \beta_r) \in \{0, 1\}^r$ there exists some $w \in [0, 1]$ such that for $q = 1, \ldots, r$

$$p_q(w) \in [0, 1/2) \quad \text{if } \beta_q = 0, \text{ and } p_q(w) \in (1/2, 1] \quad \text{if } \beta_q = 1.$$

This implies that, for each dichotomy $(S_0, S_1)$ of $S$ there is some $w \in [0, 1]$ such that for every $(i, j, k) \in S$

$$p_k(p_{j \cdot n}(p_{i \cdot n^2}(w))) < 1/2 \quad \text{if} \quad (i, j, k) \in S_0 \,,$$
$$p_k(p_{j \cdot n}(p_{i \cdot n^2}(w))) > 1/2 \quad \text{if} \quad (i, j, k) \in S_1 \,.$$

Note that $p_k(p_{j \cdot n}(p_{i \cdot n^2}(w)))$ is the value computed by $\mathcal{N}_n$ given input values $k, j, i, 1$. Therefore, choosing a suitable value for $w$, which is the parameter of $\mathcal{N}_n$, the network can induce any dichotomy on $S$. In other words, $S$ is shattered by $\mathcal{N}_n$.

It has been shown in Lemma 1 of [10] that there is an architecture $\mathcal{A}_n$ such that for each $\varepsilon > 0$ weights can be chosen for $\mathcal{A}_n$ such that the function $f_{n,\varepsilon}$ computed by this network satisfies $\lim_{\varepsilon \to 0} f_{n,\varepsilon}(y_1, \ldots, y_n, a) = y_a$. Moreover, this architecture consists of $O(n)$ computation nodes, which are linear, multiplication, and division gates. (Note that the size of $\mathcal{A}_n$ does not depend on $\varepsilon$.) Therefore, choosing $\varepsilon$ sufficiently small, we can implement the projections $\pi$ in $\mathcal{N}_n$ by networks of $O(n)$ computation nodes such that the resulting network $\mathcal{N}'_n$ still shatters $S$. Now in $\mathcal{N}'_n$ we have $O(n)$ computation nodes for implementing the three levels labeled $\Pi$ and we have in each level $P_\lambda$ a number of $O(n)$ computation nodes for computing $p_{n^{\lambda-1}}$, respectively. Assume now that the computation nodes for $p_{n^{\lambda-1}}$ can be replaced by sigmoidal networks such that on inputs from $S$ and with the parameter values defined above the resulting network $\mathcal{N}''_n$ computes the same functions as $\mathcal{N}'_n$. (Note that the computation nodes for $p_{n^{\lambda-1}}$ have no programmable parameters.)

We estimate the size of $\mathcal{N}''_n$. According to Theorem 7 of Karpinski and Macintyre [7] a sigmoidal neural network with $l$ programmable parameters and $m$ computation nodes has VC dimension $O((ml)^2)$. We have to generalize this result slightly before being able to apply it. It can readily be seen from the proof of Theorem 7 in [7] that the result also holds if the network additionally contains linear and multiplication gates. For division gates we can derive the same bound taking into account that for a gate computing division, say $x/y$, we can introduce a defining equality $x = z \cdot y$ where $z$ is a new variable. (See [7] for how to proceed.) Thus, we have that a network with $l$ programmable parameters and $m$ computation nodes, which are linear, multiplication, division, and sigmoidal gates, has VC dimension $O((ml)^2)$. In particular, if $m$ is the number of computation nodes of $\mathcal{N}''_n$, the VC dimension is $O(m^2)$. On the other hand, as we have shown above, $\mathcal{N}''_n$ can shatter a set of cardinality $n^3$. Since there are $O(n)$ sigmoidal networks in $\mathcal{N}''_n$ computing the functions $p_{n^{\lambda-1}}$, and since the number of linear, multiplication, and division gates is bounded by $O(n)$, for some value of $\lambda$ a single network computing $p_{n^{\lambda-1}}$ must have size at least $\Omega(\sqrt{n})$. This yields a lower bound of $\Omega(n^{1/4})$ for the size of a sigmoidal network computing $p_n$.

Thus far, we have assumed that the polynomials $p_n$ are computed exactly. Since polynomials are continuous functions and since we require them to be calculated only on a finite set of input values (those resulting from $S$ and from the parameter values chosen for $w$ to shatter $S$) an approximation of these polynomials is sufficient. A straightforward analysis, based on the fact that the output value of the network has a "tolerance" close to $1/2$, shows that if $p_n$ is approximated with error $O(2^{-n})$

in the $l_\infty$ norm, the resulting network still shatters the set $S$. This completes the proof of the theorem.                                                                       □

The statement of the previous theorem is restricted to the approximation of polynomials on the input domain $[0, 1]$. However, the result immediately generalizes to any arbitrary interval in $\mathbb{R}$. Moreover, it remains valid for multivariate polynomials of arbitrary input dimension.

**Corollary 2** *The approximation of polynomials of degree $k$ by sigmoidal neural networks with approximation error $O(1/k)$ in the $l_\infty$ norm requires networks of size $\Omega((\log k)^{1/4})$. This holds for polynomials over any number of variables.*

## 4   Conclusions and Open Questions

We have established lower bounds on the size of sigmoidal networks for the approximation of continuous functions. In particular, for a concrete class of polynomials we have calculated a lower bound in terms of the degree of the polynomials. The main result already holds for the approximation of univariate polynomials. Intuitively, approximation of multivariate polynomials seems to become harder when the dimension increases. Therefore, it would be interesting to have lower bounds both in terms of the degree and the input dimension.

Further, in our result the approximation error and the degree are coupled. Naturally, one would expect that the number of nodes has to grow for each fixed function when the error decreases. At present we do not know of any such lower bound.

We have not aimed at calculating the constants in the bounds. For practical applications such values are indispensable. Refining our method and using tighter results it should be straightforward to obtain such numbers. Further, we expect that better lower bounds can be obtained by considering networks of restricted depth.

To establish the result we have introduced a new method for deriving lower bounds on network sizes. One of the main arguments is to use the functions to be approximated to construct networks with large VC dimension. The method seems suitable to obtain bounds also for the approximation of other types of functions as long as they are computationally powerful enough.

Moreover, the method could be adapted to obtain lower bounds also for networks using other activation functions (e.g. more general sigmoidal functions, ridge functions, radial basis functions). This may lead to new separation results for the approximation capabilities of different types of neural networks. In order for this to be accomplished, however, an essential requirement is that small upper bounds can be calculated for the VC dimension of such networks.

### Acknowledgments

I thank Hans U. Simon for helpful discussions. This work was supported in part by the ESPRIT Working Group in Neural and Computational Learning II, Neuro-COLT2, No. 27150.

## References

[1] A. Barron. Universal approximation bounds for superposition of a sigmoidal function. *IEEE Transactions on Information Theory*, 39:930–945, 1993.

[2] C. K. Chui and X. Li. Approximation by ridge functions and neural networks with one hidden layer. *Journal of Approximation Theory*, 70:131–141, 1992.

[3] G. Cybenko. Approximation by superpositions of a sigmoidal function. *Mathematics of Control, Signals, and Systems*, 2:303–314, 1989.

[4] B. DasGupta and G. Schnitger. The power of approximating: A comparison of activation functions. In C. L. Giles, S. J. Hanson, and J. D. Cowan, editors, *Advances in Neural Information Processing Systems 5*, pages 615–622, Morgan Kaufmann, San Mateo, CA, 1993.

[5] K. Hornik. Approximation capabilities of multilayer feedforward networks. *Neural Networks*, 4:251–257, 1991.

[6] K. Hornik, M. Stinchcombe, and H. White. Multilayer feedforward networks are universal approximators. *Neural Networks*, 2:359–366, 1989.

[7] M. Karpinski and A. Macintyre. Polynomial bounds for VC dimension of sigmoidal and general Pfaffian neural networks. *Journal of Computer and System Sciences*, 54:169–176, 1997.

[8] A. G. Khovanskiĭ. *Fewnomials*, volume 88 of *Translations of Mathematical Monographs*. American Mathematical Society, Providence, RI, 1991.

[9] P. Koiran. VC dimension in circuit complexity. In *Proceedings of the 11th Annual IEEE Conference on Computational Complexity CCC'96*, pages 81–85, IEEE Computer Society Press, Los Alamitos, CA, 1996.

[10] P. Koiran and E. D. Sontag. Neural networks with quadratic VC dimension. *Journal of Computer and System Sciences*, 54:190–198, 1997.

[11] V. Y. Kreinovich. Arbitrary nonlinearity is sufficient to represent all functions by neural networks: A theorem. *Neural Networks*, 4:381–383, 1991.

[12] M. Leshno, V. Y. Lin, A. Pinkus, and S. Schocken. Multilayer feedforward networks with a nonpolynomial activation function can approximate any function. *Neural Networks*, 6:861–867, 1993.

[13] W. Maass. Noisy spiking neurons with temporal coding have more computational power than sigmoidal neurons. In M. Mozer, M. I. Jordan, and T. Petsche, editors, *Advances in Neural Information Processing Systems 9*, pages 211–217. MIT Press, Cambridge, MA, 1997.

[14] H. Mhaskar. Neural networks for optimal approximation of smooth and analytic functions. *Neural Computation*, 8:164–177, 1996.

[15] F. Scarselli and A. C. Tsoi. Universal approximation using feedforward neural networks: A survey of some existing methods and some new results. *Neural Networks*, 11:15–37, 1998.

[16] H. E. Warren. Lower bounds for approximation by nonlinear manifolds. *Transactions of the American Mathematical Society*, 133:167–178, 1968.